# A Rate Distortion Approach for Semi-Supervised Conditional Random Fields

**Yang Wang**[†][*]   **Gholamreza Haffari**[†][*]   **Shaojun Wang**[‡]   **Greg Mori**[†]

[†]School of Computing Science
Simon Fraser University
Burnaby, BC V5A 1S6, Canada
{ywang12,ghaffar1,mori}@cs.sfu.ca

[‡]Kno.e.sis Center
Wright State University
Dayton, OH 45435, USA
shaojun.wang@wright.edu

## Abstract

We propose a novel information theoretic approach for semi-supervised learning of conditional random fields that defines a training objective to combine the conditional likelihood on labeled data and the mutual information on unlabeled data. In contrast to previous minimum conditional entropy semi-supervised discriminative learning methods, our approach is grounded on a more solid foundation, the rate distortion theory in information theory. We analyze the tractability of the framework for structured prediction and present a convergent variational training algorithm to defy the combinatorial explosion of terms in the sum over label configurations. Our experimental results show the rate distortion approach outperforms standard $l_2$ regularization, minimum conditional entropy regularization as well as maximum conditional entropy regularization on both multi-class classification and sequence labeling problems.

## 1  Introduction

In most real-world machine learning problems (e.g., for text, image, audio, biological sequence data), unannotated data is abundant and can be collected at almost no cost. However, supervised machine learning techniques require large quantities of data be manually labeled so that automatic learning algorithms can build sophisticated models. Unfortunately, manual annotation of a large quantity of data is both expensive and time-consuming. The challenge is to find ways to exploit the large quantity of unlabeled data and turn it into a resource that can improve the performance of supervised machine learning algorithms. Meeting this challenge requires research at the cutting edge of automatic learning techniques, useful in many fields such as language and speech technology, image processing and computer vision, robot control and bioinformatics. A surge of semi-supervised learning research activities has occurred in recent years to devise various effective semi-supervised training schemes. Most of these semi-supervised learning algorithms are applicable only to multi-class classification problems [1, 10, 32], with very few exceptions that develop discriminative models suitable for structured prediction [2, 9, 16, 20, 21, 22].

In this paper, we propose an information theoretic approach for semi-supervised learning of conditional random fields (CRFs) [19], where we use the mutual information between the empirical distribution of unlabeled data and the discriminative model as a data-dependent regularized prior. Grandvalet and Bengio [15] and Jiao et al. [16] have proposed a similar information theoretic approach that used the conditional entropy of their discriminative models on unlabeled data as a data-dependent regularization term to obtain very encouraging results. Minimum entropy approach can be explained from data-smoothness assumption and is motivated from semi-supervised classification, using unlabeled data to enhance classification; however, its degeneracy is even more problematic and arguable by noting minimum entropy 0 can be achieved by putting all mass on one label and zeros for the rest of labels. As far as we know, there is no formal principled explanation for the validity of this minimum conditional entropy approach. Instead, our approach can be naturally cast into the rate

---

[*]These authors contributed equally to this work.

distortion theory framework which is well-known in information theory [14]. The closest work to ours is the one by Corduneanu et al. [11, 12, 13, 28]. Both works are discriminative models and do indeed use mutual information concepts. There are two major distinctions between our work and theirs. First, their approach is essentially motivated from semi-supervised classification point of view and formulated as a communication game, while our approach is based on a completely different motivation, semi-supervised clustering that uses labeled data to enhance clustering and is formulated as a data compression scheme, thus leads to a formulation distinctive from Corduneanu et al. Second, their model is non-parametric, whereas ours is parametric. As a result, their model can be trained by optimizing a convex objective function through a variant of Blahut-Arimoto alternating minimization algorithm, whereas our model is more complex and the objective function becomes non-convex. In particular, training a simple chain structured CRF model [19] in our framework turns out to be intractable even if using Blahut-Arimoto's type of alternating minimization algorithm. We develop a convergent variational approach to approximately solve this problem. Another relevant work is the information bottleneck (IB) method introduced by Tishby et al [30]. IB method is an information-theoretic framework for extracting relevant components of an input random variable $X$, with respect to an output random variable $Y$. Instead of directly compressing $X$ to its representation $Y$ subject to an expected distortion through a parametric probabilistic mapping like our proposed approach, IB method is performed by finding a third, compressed, non-parametric and model-independent representation $T$ of $X$ that is most informative about $Y$. Formally speaking, the notion of compression is quantified by the mutual information between $T$ and $X$ while the informativeness is quantified by the mutual information between $T$ and $Y$. The solutions are characterized by the bottleneck equations and can be found by a convergent re-estimation method that generalizes the Blahut-Arimoto algorithm. Finally in contrast to our approach which minimizes both the negative conditional likelihood on labeled data and the mutual information between the hidden variables and the observations on unlabeled data for a *discriminative* model, Oliver and Garg [24] have proposed maximum mutual information hidden Markov models (MMIHMM) of semi-supervised training for chain structured graph. The objective is to maximize both the joint likelihood on labeled data and the mutual information between the hidden variables and the observations on unlabeled data for a *generative* model. It is equivalent to minimizing conditional entropy of a generative HMM for the part of unlabeled data. The maximum mutual information of a generative HMM was originally proposed by Bahl et al. [4] and popularized in speech recognition community [23], but it is different from Oliver and Garg's approach in that an individual HMM is learned for each possible class (e.g., one HMM for each word string), and the point-wise mutual information between the choice of HMM and the observation sequence is maximized. It is equivalent to maximizing the conditional likelihood of a word string given observation sequence to improve the discrimination across different models [18]. Thus in essence, Bahl et al. [4] proposed a discriminative learning algorithm for generative HMMs of training utterances in speech recognition.

In the following, we first motivate our rate distortion approach for semi-supervised CRFs as a data compression scheme and formulate the semi-supervised learning paradigm as a classic rate distortion problem. We then analyze the tractability of the framework for structured prediction and present a convergent variational learning algorithm to defy the combinatorial explosion of terms in the sum over label configurations. Finally we demonstrate encouraging results with two real-world problems to show the effectiveness of the proposed approach: text categorization as a multi-class classification problem and hand-written character recognition as a sequence labeling problem. Similar ideas have been successfully applied to semi-supervised boosting [31].

## 2 Rate distortion formulation

Let $X$ be a random variable over data sequences to be labeled, and $Y$ be a random variable over corresponding label sequences. All components, $Y_i$, of $Y$ are assumed to range over a finite label alphabet $\mathcal{Y}$. Given a set of labeled examples, $\mathcal{D}^l = \left\{ (\mathbf{x}^{(1)}, \mathbf{y}^{(1)}), \cdots, (\mathbf{x}^{(N)}, \mathbf{y}^{(N)}) \right\}$, and unlabeled examples, $\mathcal{D}^u = \left\{ \mathbf{x}^{(N+1)}, \cdots, \mathbf{x}^{(M)} \right\}$, we would like to build a CRF model $p_\theta(\mathbf{y}|\mathbf{x}) = \frac{1}{Z_\theta(\mathbf{x})} \exp\left( \langle \theta, f(\mathbf{x}, \mathbf{y}) \rangle \right)$ over sequential input data $x$, where $\theta = (\theta_1, \cdots, \theta_K)^\top$, $f(\mathbf{x}, \mathbf{y}) = (f_1(\mathbf{x}, \mathbf{y}), \cdots, f_K(\mathbf{x}, \mathbf{y}))^\top$, and $Z_\theta(\mathbf{x}) = \sum_{\mathbf{y}} \exp\left( \langle \theta, f(\mathbf{x}, \mathbf{y}) \rangle \right)$. Our goal is to learn such a model from the combined set of labeled and unlabeled examples, $\mathcal{D}^l \cup \mathcal{D}^u$. For notational convenience, we assume that there are no identical examples in $\mathcal{D}^l$ and $\mathcal{D}^u$.

The standard supervised training procedure for CRFs is based on minimizing the negative log conditional likelihood of the labeled examples in $\mathcal{D}^l$

$$CL(\theta) = -\sum_{i=1}^{N} \log p_\theta(\mathbf{y}^{(i)}|\mathbf{x}^{(i)}) + \lambda U(\theta) \tag{1}$$

where $U(\theta)$ can be any standard regularizer on $\theta$, e.g. $U(\theta) = \|\theta\|^2/2$ and $\lambda$ is a parameter that controls the influence of $U(\theta)$. Regularization can alleviate over-fitting on rare features and avoid degeneracy in the case of correlated features.

Obviously, Eq. (1) ignores the unlabeled examples in $\mathcal{D}^u$. To make full use of the available training data, Grandvalet and Bengio [15] and Jiao et al. [16] proposed a semi-supervised learning algorithm that exploits a form of *minimum conditional entropy regularization* on the unlabeled data. Specifically, they proposed to minimize the following objective

$$RL_{\mathrm{minCE}}(\theta) = -\sum_{i=1}^{N} \log p_\theta(\mathbf{y}^{(i)}|\mathbf{x}^{(i)}) + \lambda U(\theta) - \gamma \sum_{j=N+1}^{M} \sum_{\mathbf{y}} p_\theta(\mathbf{y}|\mathbf{x}^{(j)}) \log p_\theta(\mathbf{y}|\mathbf{x}^{(j)}) \tag{2}$$

where the first term is the negative log conditional likelihood of the labeled data, and the third term is the conditional entropy of the CRF model on the unlabeled data. The tradeoff parameters $\lambda$ and $\gamma$ control the influences of $U(\theta)$ and the unlabeled data, respectively.

This is equivalent to minimizing the following objective (with different values of $\lambda$ and $\gamma$)

$$RL_{\mathrm{minCE}}(\theta) = D\Big(\tilde{p}_l(\mathbf{x},\mathbf{y}), \tilde{p}_l(\mathbf{x})p_\theta(\mathbf{y}|\mathbf{x})\Big) + \lambda U(\theta) + \gamma \sum_{\mathbf{x}\in\mathcal{D}^u} \tilde{p}_u(\mathbf{x})H\Big(p_\theta(\mathbf{y}|\mathbf{x})\Big) \tag{3}$$

where $D\Big(\tilde{p}_l(\mathbf{x},\mathbf{y}), \tilde{p}_l(\mathbf{x})p_\theta(\mathbf{y}|\mathbf{x})\Big) = \sum_{(\mathbf{x},\mathbf{y})\in\mathcal{D}^l} \tilde{p}_l(\mathbf{x},\mathbf{y}) \log \frac{\tilde{p}_l(\mathbf{x},\mathbf{y})}{\tilde{p}_l(\mathbf{x})p_\theta(\mathbf{y}|\mathbf{x})}$, $H\Big(p_\theta(\mathbf{y}|\mathbf{x})\Big) = \sum_{\mathbf{y}} p_\theta(\mathbf{y}|\mathbf{x}) \log p_\theta(\mathbf{y}|\mathbf{x})$. Here we use $\tilde{p}_l(\mathbf{x},\mathbf{y})$ to denote the empirical distribution of both $X$ and $Y$ on labeled data $\mathcal{D}^l$, $\tilde{p}_l(\mathbf{x})$ to denote the empirical distribution of $X$ on labeled data $\mathcal{D}^l$, and $\tilde{p}_u(\mathbf{x})$ to denote the empirical distribution of $X$ on unlabeled data $\mathcal{D}^u$.

In this paper, we propose an alternative approach for semi-supervised CRFs. Rather than using minimum conditional entropy as a regularization term on unlabeled data, we use *minimum mutual information* on unlabeled data. This approach has a nice and strong information theoretic interpretation by rate distortion theory.

We define the marginal distribution $p_\theta(\mathbf{y})$ of our discriminative model on unlabeled data $\mathcal{D}^u$ to be $p_\theta(\mathbf{y}) = \sum_{\mathbf{x}\in\mathcal{D}^u} \tilde{p}_u(\mathbf{x})p_\theta(\mathbf{y}|\mathbf{x})$ over the input data $\mathbf{x}$. Then the mutual information between the empirical distribution $\tilde{p}(\mathbf{x})$ and the discriminative model is

$$I\Big(\tilde{p}_u(\mathbf{x}), p_\theta(\mathbf{y}|\mathbf{x})\Big) = \sum_{\mathbf{x}\in\mathcal{D}^u}\sum_{\mathbf{y}} \tilde{p}_u(\mathbf{x})p_\theta(\mathbf{y}|\mathbf{x}) \log\Big(\frac{\tilde{p}_u(\mathbf{x})p_\theta(\mathbf{y}|\mathbf{x})}{\tilde{p}_u(\mathbf{x})p_\theta(\mathbf{y})}\Big) = H\Big(p_\theta(\mathbf{y})\Big) - \sum_{\mathbf{x}\in\mathcal{D}^u} \tilde{p}_u(\mathbf{x})H\Big(p_\theta(\mathbf{y}|\mathbf{x})\Big)$$

where $H\Big(p_\theta(\mathbf{y})\Big) = -\sum_{\mathbf{y}}\sum_{\mathbf{x}\in\mathcal{D}^u} \tilde{p}_u(\mathbf{x})p_\theta(\mathbf{y}|\mathbf{x}) \log\Big(\sum_{\mathbf{x}\in\mathcal{D}^u} \tilde{p}_u(\mathbf{x})p_\theta(\mathbf{y}|\mathbf{x})\Big)$ is the entropy of the label $Y$ on unlabeled data. Thus in rate distortion terminology, the empirical distribution of unlabeled data $\tilde{p}_u(\mathbf{x})$ corresponds to input distribution, the model $p_\theta(\mathbf{y}|\mathbf{x})$ corresponds to the probabilistic mapping from $X$ to $Y$, and $p_\theta(\mathbf{y})$ corresponds to the output distribution of $Y$.

Our proposed rate distortion approach for semi-supervised CRFs optimizes the following constrained optimization problem,

$$\min_{\theta} \ I\Big(\tilde{p}_u(\mathbf{x}), p_\theta(\mathbf{y}|\mathbf{x})\Big) \ \text{ s.t. } \ D\Big(\tilde{p}_l(\mathbf{x},\mathbf{y}), \tilde{p}_l(\mathbf{x})p_\theta(\mathbf{y}|\mathbf{x})\Big) + \lambda U(\theta) \le d \tag{4}$$

The rationale for this formulation can be seen from an information-theoretic perspective using the rate distortion theory [14]. Assume we have a source $X$ with a source distribution $p(\mathbf{x})$ and its compressed representation $Y$ through a probabilistic mapping $p_\theta(\mathbf{y}|\mathbf{x})$. If there is a large set of features (infinite in the extreme case), this probabilistic mapping might be too redundant. We'd better look for its minimum description. What determines the quality of the compression is the information rate, i.e. the average number of bits per message needed to specify an element in the representation without confusion. According to the standard asymptotic arguments [14], this quantity is bounded below by the mutual information $I\Big(p(\mathbf{x}), p_\theta(\mathbf{y}|\mathbf{x})\Big)$ since the average cardinality of the partitioning of $X$ is given by the ratio of the volume of $X$ to the average volume of the elements of $X$

that are mapped to the same representation $Y$ through $p_\theta(\mathbf{y}|\mathbf{x})$, $2^{H(X)}/2^{H(X|Y)} = 2^{I(X,Y)}$. Thus mutual information is the minimum information rate and is used as a good metric for clustering [26, 27]. True distribution of $X$ should be used to compute the mutual information. Since it is unknown, we use its empirical distribution on unlabeled data set $\mathcal{D}^u$ and the mutual information $I\big(\tilde{p}_u(\mathbf{x}), p_\theta(\mathbf{y}|\mathbf{x})\big)$ instead. However, information rate alone is not enough to characterize good representation since the rate can always be reduced by throwing away many features in the probabilistic mapping. This makes the mapping likely too simple and leads to distortion. Therefore we need an additional constraint provided through a distortion function which is presumed to be small for good representations. Apparently there is a tradeoff between minimum representation and maximum distortion. Since joint distribution gives the distribution for the pair of $X$ and its representation $Y$, we choose the log likelihood ratio, $\log \frac{p(\mathbf{x},\mathbf{y})}{p(\mathbf{x})p_\theta(\mathbf{y}|\mathbf{x})}$, plus a regularized complexity term of $\theta$, $\lambda U(\theta)$, as the distortion function. Thus the expected distortion is the non-negative term $D\big(p(\mathbf{x},\mathbf{y}), p(\mathbf{x})p_\theta(\mathbf{y}|\mathbf{x})\big) + \lambda U(\theta)$. Again true distributions $p(\mathbf{x},\mathbf{y})$ and $p(\mathbf{x})$ should be used here, but they are unknown. In semi-supervised setting, we have labeled data available which provides valuable information to measure the distortion: we use the empirical distributions on labeled data set $\mathcal{D}^l$ and the expected distortion $D\big(\tilde{p}_l(\mathbf{x},\mathbf{y}), \tilde{p}_l(\mathbf{x})p_\theta(\mathbf{y}|\mathbf{x})\big) + \lambda U(\theta)$ instead to encode the information provided by labeled data, and add a distortion constraint we should respect for data compression to help the clustering. There is a monotonic tradeoff between the rate of the compression and the expected distortion: the larger the rate, the smaller is the achievable distortion. Given a distortion measure between $X$ and $Y$ on the labeled data set $\mathcal{D}^l$, what is the minimum rate description required to achieve a particular distortion on the unlabeled data set $\mathcal{D}^u$? The answer can be obtained by solving (4).

Following standard procedure, we convert the constrained optimization problem (4) into an unconstrained optimization problem which minimizes the following objective:

$$RL_{\mathrm{MI}}(\theta) = I\big(\tilde{p}_u(\mathbf{x}), p_\theta(\mathbf{y}|\mathbf{x})\big) + \kappa\Big(D\big(\tilde{p}_l(\mathbf{x},\mathbf{y}), \tilde{p}_l(\mathbf{x})p_\theta(\mathbf{y}|\mathbf{x})\big) + \lambda U(\theta)\Big) \tag{5}$$

where $\kappa > 0$, which again is equivalent to minimizing the following objective (with $\gamma = \frac{1}{\kappa}$)[1]:

$$RL_{\mathrm{MI}}(\theta) = D\big(\tilde{p}_l(\mathbf{x},\mathbf{y}), \tilde{p}_l(\mathbf{x})p_\theta(\mathbf{y}|\mathbf{x})\big) + \lambda U(\theta) + \gamma I\big(\tilde{p}_u(\mathbf{x}), p_\theta(\mathbf{y}|\mathbf{x})\big) \tag{6}$$

If (4) is a convex optimization problem, then for every solution $\theta$ to Eq. (4) found using some particular value of $d$, there is some corresponding value of $\gamma$ in the optimization problem (6) that will give the same $\theta$. Thus, these are two equivalent re-parameterizations of the same problem. The equivalence between the two problems can be verified using convex analysis [8] by noting that the Lagrangian for the constrained optimization (4) is exactly the objective in the optimization (5) (plus a constant that does not depend on $\theta$), where $\kappa$ is the Lagrange multiplier. Thus, (4) can be solved by solving either (5) or (6) for an appropriate $\kappa$ or $\gamma$. Unfortunately (4) is not a convex optimization problem, because its objective $I\big(\tilde{p}_u(\mathbf{x}), p_\theta(\mathbf{y}|\mathbf{x})\big)$ is not convex. This can be verified using the same argument as in the minimum conditional entropy regularization case [15, 16]. There may be some minima of (4) that do not minimize (5) or (6) whatever the value of $\kappa$ or $\gamma$ may be. This is however not essential to motivate the optimization criterion. Moreover there are generally local minima in (5) or (6) due to the non-convexity of its mutual information regularization term.

Another training method for semi-supervised CRFs is the *maximum entropy* approach, maximizing conditional entropy (minimizing negative conditional entropy) over unlabeled data $\mathcal{D}^u$ subject to the constraint on labeled data $\mathcal{D}^l$,

$$\min_\theta \Big(-\sum_{\mathbf{x}\in\mathcal{D}^u} \tilde{p}_u(\mathbf{x})H\big(p_\theta(\mathbf{y}|\mathbf{x})\big)\Big) \quad \text{s.t.} \quad D\big(\tilde{p}_l(\mathbf{x},\mathbf{y}), \tilde{p}_l(\mathbf{x})p_\theta(\mathbf{y}|\mathbf{x})\big) + \lambda U(\theta) \leq d \tag{7}$$

again following standard procedure, we convert the constrained optimization problem (7) into an unconstrained optimization problem which minimizes the following objective:

$$RL_{\mathrm{maxCE}}(\theta) = D\big(\tilde{p}_l(\mathbf{x},\mathbf{y}), \tilde{p}_l(\mathbf{x})p_\theta(\mathbf{y}|\mathbf{x})\big) + \lambda U(\theta) - \gamma \sum_{\mathbf{x}\in\mathcal{D}^u} \tilde{p}_u(\mathbf{x})H\big(p_\theta(\mathbf{y}|\mathbf{x})\big) \tag{8}$$

Again minimizing (8) is not exactly equivalent to (7); however, it is not essential to motivate the optimization criterion. When comparing maximum entropy approach with minimum conditional entropy approach, there is only a sign change on conditional entropy term.

For non-parametric models, using the analysis developed in [5, 6, 7, 25], it can be shown that maximum conditional entropy approach is equivalent to rate distortion approach when we compress code vectors in a mass constrained scheme [25]. But for parametric models such as CRFs, these three approaches are completely distinct.

The difference between our rate distortion approach for semi-supervised CRFs (6) and the minimum conditional entropy regularized semi-supervised CRFs (2) is not only on the different sign of conditional entropy on unlabeled data but also the additional term – entropy of $p_\theta(\mathbf{y})$ on unlabeled data. It is this term that makes direct computation of the derivative of the objective for the rate distortion approach for semi-supervised CRFs intractable. To see why, we take derivative of this term with respect to $\theta$, we have:

$$
\frac{\partial}{\partial \theta}\Big(-H(p_\theta(\mathbf{y}))\Big) = \sum_{\mathbf{x}\in\mathcal{D}^u} \tilde{p}_u(\mathbf{x}) \sum_{\mathbf{y}} p_\theta(\mathbf{y}|\mathbf{x}) f(\mathbf{x},\mathbf{y}) \log\Big(\sum_{\mathbf{x}\in\mathcal{D}^u} \tilde{p}_u(\mathbf{x}) p_\theta(\mathbf{y}|\mathbf{x})\Big)
$$
$$
- \sum_{\mathbf{x}\in\mathcal{D}^u} \tilde{p}_u(\mathbf{x}) \sum_{\mathbf{y}} p_\theta(\mathbf{y}|\mathbf{x}) \log\Big(\sum_{\mathbf{x}\in\mathcal{D}^u} \tilde{p}_u(\mathbf{x}) p_\theta(\mathbf{y}|\mathbf{x})\Big) \sum_{\mathbf{y}'} p_\theta(\mathbf{y}'|\mathbf{x}) f(\mathbf{x},\mathbf{y}')
$$

In the case of structured prediction, the number of sums over $Y$ is exponential, and there is a sum inside the $\log$. These make the computation of the derivative intractable even for a simple chain structured CRF.

An alternative way to solve (6) is to use the famous algorithm for the computation of the rate distortion function established by Blahut [6] and Arimoto [3]. Corduneanu and Jaakkola [12, 13] proposed a distributed propagation algorithm, a variant of Blahut-Arimoto algorithm, to solve their problem. However as illustrated in the following, this approach is still intractable for structured prediction in our case.

By extending a lemma for computing rate distortion in [14] to parametric models, we can rewrite the minimization problem (5) of mutual information regularized semi-supervised CRFs as a double minimization,

$$
\min_{\theta} \min_{r(\mathbf{y})} g(\theta, r(\mathbf{y})) \quad \text{where}
$$

$$
g(\theta, r(\mathbf{y})) = \sum_{\mathbf{x}\in\mathcal{D}^u} \sum_{\mathbf{y}} \tilde{p}_u(\mathbf{x}) p_\theta(\mathbf{y}|\mathbf{x}) \log \frac{p_\theta(\mathbf{y}|\mathbf{x})}{r(\mathbf{y})} + \kappa\Big(D\big(\tilde{p}_l(\mathbf{x},\mathbf{y}), \tilde{p}_l(\mathbf{x})p_\theta(\mathbf{y}|\mathbf{x})\big) + \lambda U(\theta)\Big)
$$

We can use an alternating minimization algorithm to find a local minimum of $RL_{MI}(\theta)$. First, we assign the initial CRF model to be the optimal solution of the supervised CRF on labeled data and denote it as $p_{\theta^{(0)}}(\mathbf{y}|\mathbf{x})$. Then we define $r^{(0)}(\mathbf{y})$ and in general $r^{(t)}(\mathbf{y})$ for $t \geq 1$ by

$$
r^{(t)}(\mathbf{y}) = \sum_{x\in\mathcal{D}^u} \tilde{p}_u(\mathbf{x}) p_{\theta^{(t)}}(\mathbf{y}|\mathbf{x}) \tag{9}
$$

In order to define $p_{\theta^{(1)}}(\mathbf{y}|\mathbf{x})$ and in general $p_{\theta^{(t)}}(\mathbf{y}|\mathbf{x})$, we need to find the $p_\theta(\mathbf{y}|\mathbf{x})$ which minimizes $g$ for a given $r(\mathbf{y})$. The gradient of $g(\theta, r(\mathbf{y}))$ with respect to $\theta$ is

$$
\frac{\partial}{\partial \theta} g(\theta, r(\mathbf{y})) = \sum_{i=N+1}^{M} \tilde{p}_u(\mathbf{x}^{(i)}) \Big(\mathrm{cov}_{p_\theta(\mathbf{y}|\mathbf{x}^{(i)})}\big[f(\mathbf{x}^{(i)},\mathbf{y})\big]\theta - \sum_{\mathbf{y}} p_\theta(\mathbf{y}|\mathbf{x}^{(i)}) f(\mathbf{x}^{(i)},\mathbf{y}) \log r(\mathbf{y}) \tag{10}
$$

$$
+ \sum_{\mathbf{y}} p_\theta(\mathbf{y}|\mathbf{x}^{(i)}) \log r(\mathbf{y}) \sum_{\mathbf{y}'} p_\theta(\mathbf{y}'|\mathbf{x}^{(i)}) f(\mathbf{x}^{(i)},\mathbf{y}')\Big) \tag{11}
$$

$$
- \kappa \sum_{i=1}^{N} \tilde{p}_l(\mathbf{x}^{(i)}) \Big(f(\mathbf{x}^{(i)},\mathbf{y}^{(i)}) - \sum_{\mathbf{y}} p_\theta(\mathbf{y}|\mathbf{x}^{(i)}) f(\mathbf{x}^{(i)},\mathbf{y})\Big) + \kappa\lambda \frac{\partial}{\partial \theta} U(\theta) \tag{12}
$$

Even though the first term in Eq. (10) and (12) can be efficiently computed via recursive formulas [16], we run into the same intractable problem to compute the second term Eq. (10) and Eq. 11) since the number of sums over $Y$ is exponential and implicitly there is a sum inside the $\log$ due to $r(\mathbf{y})$. This makes the computation of the derivative in the alternating minimization algorithm intractable.

# 3 A variational training procedure

In this section, we derive a convergent variational algorithm to train rate distortion based semi-supervised CRFs for sequence labeling. The basic idea of convexity-based variational inference is to make use of Jensen's inequality to obtain an adjustable upper bound on the objective function [17]. Essentially, one considers a family of upper bounds indexed by a set of variational parameters. The variational parameters are chosen by an optimization procedure that attempts to find the tightest possible upper bound.

Following Jordan et al. [17], we begin by introducing a variational distribution $q(x)$ to bound $H(p_\theta(\mathbf{y}))$ using Jensen's inequality as the following,

$$
\begin{aligned}
H(p_\theta(\mathbf{y})) &= -\sum_{\mathbf{y}} \sum_{\mathbf{x} \in \mathcal{D}^u} \tilde{p}_u(\mathbf{x}) p_\theta(\mathbf{y}|\mathbf{x}) \log \left( \sum_{\mathbf{x} \in \mathcal{D}^u} \frac{\tilde{p}_u(\mathbf{x}) p_\theta(\mathbf{y}|\mathbf{x})}{q(\mathbf{x})} q(\mathbf{x}) \right) \\
&\leq -\sum_{\mathbf{y}} \sum_{j=N+1}^{M} \tilde{p}_u(\mathbf{x}^{(j)}) p_\theta(\mathbf{y}|\mathbf{x}^{(j)}) \left[ \sum_{l=N+1}^{M} q(\mathbf{x}^{(l)}) \log \left( \frac{\tilde{p}_u(\mathbf{x}^{(l)}) p_\theta(\mathbf{y}|\mathbf{x}^{(l)})}{q(\mathbf{x}^{(l)})} \right) \right]
\end{aligned}
$$

Thus the desideratum of finding a tight upper bound of $RL_{\mathrm{MI}}(\theta)$ in Eq. (6) translates directly into the following alternative optimization problem:

$$
(\theta^*, q^*) = \min_{\theta, q} \mathcal{U}(\theta, q) \qquad \text{where}
$$

$$
\mathcal{U}(\theta, q) =
$$

$$
-\sum_{i=1}^{N} \tilde{p}_l(\mathbf{x}^{(i)}) \log p_\theta(\mathbf{y}^{(i)}|\mathbf{x}^{(i)}) + \lambda U(\theta) - \gamma \sum_{j=N+1}^{M} \sum_{l=N+1}^{M} \tilde{p}_u(\mathbf{x}^{(j)}) q(\mathbf{x}^{(l)}) \sum_{\mathbf{y}} p_\theta(\mathbf{y}|\mathbf{x}^{(j)}) \log p_\theta(\mathbf{y}|\mathbf{x}^{(l)}) \quad (13)
$$

$$
-\gamma \sum_{j=N+1}^{M} \tilde{p}_u(\mathbf{x}^{(j)}) \sum_{l=N+1}^{M} q(\mathbf{x}^{(l)}) \log \frac{\tilde{p}_u(\mathbf{x}^{(l)})}{q(\mathbf{x}^{(l)})} + \gamma \sum_{j=N+1}^{M} \sum_{\mathbf{y}} \tilde{p}_u(\mathbf{x}^{(j)}) p_\theta(\mathbf{y}|\mathbf{x}^{(j)}) \log p_\theta(\mathbf{y}|\mathbf{x}^{(j)}) \quad (14)
$$

Minimizing $\mathcal{U}$ with respect to $q$ has a closed form solution,

$$
q(\mathbf{x}^{(l)}) = \frac{\tilde{p}_u(\mathbf{x}^{(l)}) \exp \left( \sum_{j=N+1}^{M} \sum_{\mathbf{y}} \tilde{p}_u(\mathbf{x}^{(j)}) p_\theta(\mathbf{y}|\mathbf{x}^{(j)}) \log p_\theta(\mathbf{y}|\mathbf{x}^{(l)}) \right)}{\sum_{k=1}^{M} \tilde{p}_u(\mathbf{x}^{(k)}) \exp \left( \sum_{j=N+1}^{M} \sum_{\mathbf{y}} \tilde{p}_u(\mathbf{x}^{(j)}) p_\theta(\mathbf{y}|\mathbf{x}^{(j)}) \log p_\theta(\mathbf{y}|\mathbf{x}^{(k)}) \right)} \quad \forall \, \mathbf{x}^{(l)} \in \mathcal{D}^u \quad (15)
$$

It can be shown that

$$
\mathcal{U}(\theta, q) \geq RL_{\mathrm{MI}}(\theta) + \sum_{\mathbf{y}} \sum_{\mathbf{x} \in \mathcal{D}^u} \tilde{p}_u(\mathbf{x}) p_\theta(\mathbf{y}|\mathbf{x}) \sum_{\mathbf{x} \in \mathcal{D}^u} D\Big(q(\mathbf{x}), q_\theta(\mathbf{x}|\mathbf{y})\Big) \geq 0 \quad (16)
$$

where $q_\theta(\mathbf{x}|\mathbf{y}) = \frac{\tilde{p}_u(\mathbf{x}) p_\theta(\mathbf{y}|\mathbf{x})}{\sum_{\mathbf{x} \in \mathcal{D}^u} \tilde{p}_u(\mathbf{x}) p_\theta(\mathbf{y}|\mathbf{x})}$ $\forall \, \mathbf{x} \in \mathcal{D}^u$. Thus $\mathcal{U}$ is bounded below, the alternative minimization algorithm monotonically decreases $\mathcal{U}$ and converges.

In order to calculate the derivative of $\mathcal{U}$ with respect to $\theta$, we just need to notice that the first term in Eq. (13) is the log-likelihood in CRF, and the first term in Eq. (14) is a constant and second term in Eq. (14) is the conditional entropy in [16]. They all can be efficiently computed [16, 21]. In the following, we show how to compute the derivative of the last term in Eq.(13) using an idea similar to that proposed in [21]. Without loss of generality, we assume all the unlabeled data are of equal lengths in the sequence labeling case. We will describe how to handle the case of unequal lengths in Sec. 4.

If we define $\mathcal{A}(\mathbf{y}, \mathbf{x}^{(j)}, \mathbf{x}^{(l)}) = \sum_{\mathbf{y}} p_\theta(\mathbf{y}|\mathbf{x}^{(j)}) \log p_\theta(\mathbf{y}|\mathbf{x}^{(l)})$ in (13) for a fixed $(j, l)$ pair, where we assume $\mathbf{x}^{(j)}$ and $\mathbf{x}^{(l)}$ form two linear-chain graphs of equal lengths, we can calculate the derivative of $\mathcal{A}(\mathbf{y}, \mathbf{x}^{(j)}, \mathbf{x}^{(l)})$ with respect to the $k$-th parameter $\theta_k$, where all the terms can be computed through standard dynamic programming techniques in CRFs except one term $\sum_{\mathbf{y}} p_\theta(\mathbf{y}|\mathbf{x}^{(j)}) \log p_\theta(\mathbf{y}|\mathbf{x}^{(l)}) f_k(\mathbf{x}^{(j)}, \mathbf{y})$. Nevertheless similar to [21], we compute this term as follows [21]: we first define *pairwise subsequence constrained entropy* on $(\mathbf{x}^{(j)}, \mathbf{x}^{(l)})$ (as suppose to the *subsequence constrained entropy* defined in [21]) as:

$$
H_{jl}^\sigma(\mathbf{y}_{-(a..b)}|y_{a..b}, \mathbf{x}^{(j)}, \mathbf{x}^{(l)}) = \sum_{\mathbf{y}_{-(a..b)}} p_\theta(\mathbf{y}_{-(a..b)}|y_{a..b}, x^{(j)}) \log p_\theta(\mathbf{y}_{-(a..b)}|y_{a..b}, x^{(l)})
$$

where $\mathbf{y}_{-(a..b)}$ is the label sequence with its subsequence $y_{a..b}$ fixed. If we have $H_{jl}^\sigma$ for all $(a, b)$, then the term $\sum_{\mathbf{y}} p_\theta(\mathbf{y}|\mathbf{x}^{(j)}) \log p_\theta(\mathbf{y}|\mathbf{x}^{(l)}) f_k(\mathbf{x}^{(j)}, \mathbf{y})$ can be easily computed. Using the independence property of linear-chain CRF, we have the following:

$$
\sum_{\mathbf{y}_{-(a..b)}} p_\theta(\mathbf{y}_{-(a..b)}, y_{a..b}|\mathbf{x}^{(j)}) \log p_\theta(\mathbf{y}_{-(a..b)}, y_{a..b}|\mathbf{x}^{(l)})
$$
$$
= \quad p_\theta(y_{a..b}|\mathbf{x}^{(j)}) \log p_\theta(y_{a..b}|\mathbf{x}^{(l)}) + p_\theta(y_{a..b}|\mathbf{x}^{(j)}) H_{jl}^\alpha(\mathbf{y}_{1..(a-1)}|y_a, \mathbf{x}^{(j)}, \mathbf{x}^{(l)})
$$
$$
+ p_\theta(y_{a..b}|\mathbf{x}^{(j)}) H_{jl}^\beta(\mathbf{y}_{(b+1)..n}|y_b, \mathbf{x}^{(j)}, \mathbf{x}^{(l)})
$$

Given $H_{jl}^\alpha(\cdot)$ and $H_{jl}^\beta(\cdot)$, any sequence entropy can be computed in constant time [21]. Computing $H_{jl}^\alpha(\cdot)$ can be done using the following dynamic programming [21]:

$$
H_{jl}^\alpha(\mathbf{y}_{1..i}|y_{i+1}, \mathbf{x}^{(j)}, \mathbf{x}^{(l)}) \quad = \quad \sum_{y_i} p_\theta(y_i|y_{i+1}, \mathbf{x}^{(j)}) \log p_\theta(y_i|y_{i+1}, \mathbf{x}^{(l)})
$$
$$
+ \sum_{y_i} p_\theta(y_i|y_{i+1}, \mathbf{x}^{(j)}) H_{jl}^\alpha(\mathbf{y}_{1..(i-1)}|y_i, \mathbf{x}^{(j)}, \mathbf{x}^{(l)})
$$

The base case for the dynamic programming is $H_{jl}^\alpha(\emptyset|y_1, \mathbf{x}^{(j)}, \mathbf{x}^{(l)}) = 0$. All the probabilities (i.e., $p_\theta(y_i|y_{i+1}, \mathbf{x}^j)$) needed in the above formula can be obtained using belief propagation. $H_{jl}^\beta(\cdot)$ can be similarly computed using dynamic programming.

## 4   Experiments

We compare our rate distortion approach for semi-supervised learning with one of the state-of-the-art semi-supervised learning algorithms, minimum conditional entropy approach and maximum conditional entropy approach on two real-world problems: text categorization and hand-written character recognition. The purpose of the first task is to show the effectiveness of rate distortion approach over minimum and maximum conditional entropy approaches when no approximation is needed in training. In the second task, a variational method has to be used to train semi-supervised chain structured CRFs. We demonstrate the effectiveness of the rate distortion approach over minimum and maximum conditional entropy approaches even when an approximation is used during training.

### 4.1   Text categorization

We select different class pairs from the 20 newsgroup dataset [2] to construct our binary classification problems. The chosen classes are similar to each other and thus hard for classification algorithms. We use Porter stemmer to reduce the morphological word forms. For each label, we rank words based on their mutual information with that label (whether it predicts label 1 or 0). Then we choose the top 100 words as our features. For each problem, we select 15% of the training data, almost 150 instances, as the labeled training data and select the unlabeled data from the remaining data. The validation set (for setting the free parameters, e.g. $\lambda$ and $\gamma$) contains 100 instances. The test set contains about 700 instances. We vary the ratio between the amount of unlabeled and labeled data, repeat the experiments ten times with different randomly selected labeled and unlabeled training data, and report the mean and standard deviation over different trials. For each run, we initialize the model parameter for mutual information (MI) regularization and maximum/minimum conditional entropy (CE) regularization using the parameter learned from a $l_2$-regularized logistic regression classifier. Figure 1 shows the classification accuracies of these four regularization methods versus the ratio between the amount of unlabeled and labeled data on different classification problems. We can see that mutual information regularization outperforms the other three regularization schemes. In most cases, maximum CE regularization outperforms minimum CE regularization and the baseline (logistic regression with $l_2$ regularization) which uses only the labeled data. Although the randomly selected labeled instances are different for different experiments, we should not see a significant difference in the performance of the learned models based on the baseline; since for each particular ratio of labeled and unlabeled data, the performance is averaged over ten runs. We suspect the reason for the performance differences of the baselines models in Figure 1 is due to our feature selection phase.

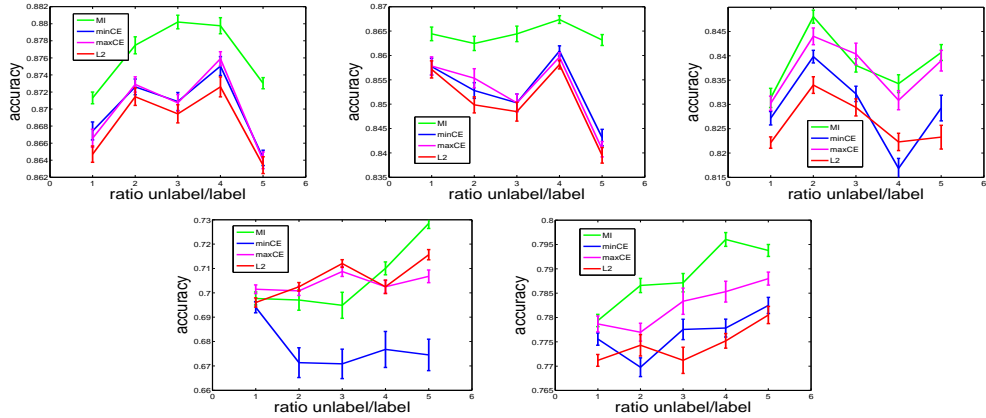

Figure 1: *Results on five different binary classification problems in text categorization (left to right): comp.os.ms-windows.misc vs comp.sys.mac.hardware; rec.autos vs rec.motorcycles; rec.sport.baseball vs rec.sport.hockey; talk.politics.guns vs talk.politics.misc; sci.electronics vs sci.med.*

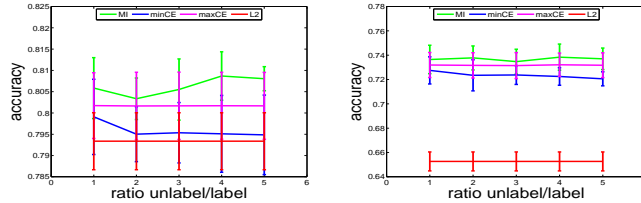

Figure 2: *Results on hand-written character recognition: (left) sequence labeling; (right) multi-class classification.*

## 4.2   Hand-written character recognition

Our dataset for hand-written character recognition contains ∼6000 handwritten words with average length of ∼8 characters. Each word was divided into characters, each character is resized to a $16 \times 8$ binary image. We choose ∼600 words as labeled data, ∼600 words as validation data, ∼2000 words as test data. Similar to text categorization, we vary the ratio between the amount of unlabeled and labeled data, and report the mean and standard deviation of classification accuracies over several trials.

We use a chain structured graph to model hand-written character recognition as a sequence labeling problem, similar to [29]. Since the unlabeled data may have different lengths, we modify the mutual information as $I = \sum_\ell I_\ell$, where $I_\ell$ is the mutual information computed on all the unlabeled data with length $\ell$. We compare our approach (MI) with other regularizations (maximum/minimum conditional entropy, $l_2$). The results are shown in Fig. 2 (left). As a sanity check, we have also tried solving hand-written character recognition as a multi-class classification problem, i.e. without considering the correlation between adjacent characters in a word. The results are shown in Fig. 2 (right). We can see that MI regularization outperforms maxCE, minCE and $l_2$ regularizations in both multi-class and sequence labeling cases. There are significant gains in the structured learning compared with the standard multi-class classification setting.

## 5   Conclusion and future work

We have presented a new semi-supervised discriminative learning algorithm to train CRFs. The proposed approach is motivated by the rate distortion framework in information theory and utilizes the mutual information on the unlabeled data as a regularization term, to be more precise a data dependent prior. Even though a variational approximation has to be used during training process for even a simple chain structured graph, our experimental results show that our proposed rate distortion approach outperforms supervised CRFs with $l_2$ regularization and a state-of-the-art semi-supervised minimum conditional entropy approach as well as semi-supervised maximum conditional entropy approach in both multi-class classification and sequence labeling problems. As future work, we would like to apply this approach to other graph structures, develop more efficient learning algorithms and illuminate how reducing the information rate helps generalization.

## Footnotes

[1]For the part of unlabeled data, the MMIHMM algorithm [24] maximizes mutual information, $I(\tilde{p}_u(\mathbf{x}), p_\theta(\mathbf{x}|\mathbf{y}))$, of a generative model $p_\theta(x|y)$ instead, which is equivalent to minimizing conditional entropy of a generative model $p_\theta(x|y)$, since $I(\tilde{p}_u(\mathbf{x}), p_\theta(\mathbf{x}|\mathbf{y})) = H(\tilde{p}_u(\mathbf{x})) - H(p_\theta(\mathbf{x}|\mathbf{y}))$ and $H(\tilde{p}_u(\mathbf{x}))$ is a constant.

[2]http://people.csail.mit.edu/jrennie/20Newsgroups.

# References

[1] S. Abney. *Semi-Supervised Learning for Computational Linguistics*. Chapman & Hall/CRC, 2007.

[2] Y. Altun, D. McAllester and M. Belkin. Maximum margin semi-supervised learning for structured variables. *NIPS* 18:33-40, 2005.

[3] S. Arimoto. An algorithm for computing the capacity of arbitrary discrete memoryless channels. *IEEE Transactions on Information Theory*, 18:1814-1820, 1972.

[4] L. Bahl, P. Brown, P. de Souza and R. Mercer. Maximum mutual information estimation of hidden Markov model parameters for speech recognition. *ICASSP*, 11:49-52, 1986.

[5] T. Berger and J. Gibson. Lossy source coding. *IEEE Transactions on Information Theory*, 44(6):2693-2723, 1998.

[6] R. Blahut. Computation of channel capacity and rate-distortion functions. *IEEE Transactions on Information Theory*, 18:460-473, 1972.

[7] R. Blahut. *Principles and Practice of Information Theory*, Addison-Wesley, 1987.

[8] S. Boyd and L. Vandenberghe. *Convex Optimization*, Cambridge University Press, 2004.

[9] U. Brefeld and T. Scheffer. Semi-supervised learning for structured output variables. *ICML*, 145-152, 2006.

[10] O. Chapelle, B. Scholköpf and A. Zien. *Semi-Supervised Learning*, MIT Press, 2006.

[11] A. Corduneanu and T. Jaakkola. On information regularization. *UAI*, 151-158, 2003.

[12] A. Corduneanu and T. Jaakkola. Distributed information regularization on graphs. *NIPS*, 17:297-304, 2004.

[13] A. Corduneanu and T. Jaakkola. Data dependent regularization. In *Semi-Supervised Learning*, O. Chapelle, B. Scholköpf and A. Zien, (Editors), 163-182, MIT Press, 2006.

[14] T. Cover and J. Thomas. *Elements of Information Theory*, Wiley, 1991.

[15] Y. Grandvalet and Y. Bengio. Semi-supervised learning by entropy minimization. *NIPS*, 17:529-536, 2004.

[16] F. Jiao, S. Wang, C. Lee, R. Greiner and D. Schuurmans. Semi-supervised conditional random fields for improved sequence segmentation and labeling. *COLING/ACL*, 209-216, 2006.

[17] M. Jordan, Z. Ghahramani, T. Jaakkola and L. Saul. Introduction to variational methods for graphical models. *Machine Learning*, 37:183-233, 1999.

[18] D. Jurafsky and J. Martin. *Speech and Language Processing*, 2nd Edition, Prentice Hall, 2008.

[19] J. Lafferty, A. McCallum and F. Pereira. Conditional random fields: Probabilistic models for segmenting and labeling sequence data. *ICML*, 282-289, 2001.

[20] C. Lee, S. Wang, F. Jiao, D. Schuurmans and R. Greiner. Learning to model spatial dependency: Semi-supervised discriminative random fields. *NIPS*, 19:793-800, 2006.

[21] G. Mann and A. McCallum. Efficient computation of entropy gradient for semi-supervised conditional random fields. *NAACL/HLT*, 109-112, 2007.

[22] G. Mann and A. McCallum. Generalized expectation criteria for semi-supervised learning of conditional random fields. *ACL*, 870-878, 2008.

[23] Y. Normandin. Maximum mutual information estimation of hidden Markov models. In *Automatic Speech and Speaker Recognition: Advanced Topics*, C. Lee, F. Soong and K. Paliwal (Editors), 57-81, Springer, 1996.

[24] N. Oliver and A. Garg. MMIHMM: maximum mutual information hidden Markov models. *ICML*, 466-473, 2002.

[25] K. Rose. Deterministic annealing for clustering, compression, classification, regression, and related optimization problems. *Proceedings of the IEEE*, 80:2210-2239, 1998.

[26] N. Slonim, G. Atwal, G. Tkacik and W. Bialek. Information based clustering. *Proceedings of National Academy of Science (PNAS)*, 102:18297-18302, 2005.

[27] S. Still and W. Bialek. How many clusters? An information theoretic perspective. *Neural Computation*, 16:2483-2506, 2004.

[28] M. Szummer and T. Jaakkola. Information regularization with partially labeled data. *NIPS*, 1025-1032, 2002.

[29] B. Taskar, C. Guestrain and D. Koller. Max-margin Markov networks. *NIPS*, 16:25-32, 2003.

[30] N. Tishby, F. Pereira, and W. Bialek. The information bottleneck method. *The 37th Annual Allerton Conference on Communication, Control, and Computing*, 368-377, 1999.

[31] L. Zheng, S. Wang, Y. Liu and C. Lee. Information theoretic regularization for semi-supervised boosting. *KDD*, 1017-1026, 2009.

[32] X. Zhu. Semi-supervised learning literature survey. *Computer Sciences TR 1530*, University of Wisconsin Madison, 2007.

